# Discriminatively Trained Sparse Code Gradients for Contour Detection

**Xiaofeng Ren and Liefeng Bo**
Intel Science and Technology Center for Pervasive Computing, Intel Labs
Seattle, WA 98195, USA
{xiaofeng.ren,liefeng.bo}@intel.com

## Abstract

Finding contours in natural images is a fundamental problem that serves as the basis of many tasks such as image segmentation and object recognition. At the core of contour detection technologies are a set of hand-designed gradient features, used by most approaches including the state-of-the-art *Global Pb* (gPb) operator. In this work, we show that contour detection accuracy can be significantly improved by computing *Sparse Code Gradients* (SCG), which measure contrast using patch representations automatically learned through sparse coding. We use *K-SVD* for dictionary learning and *Orthogonal Matching Pursuit* for computing sparse codes on oriented local neighborhoods, and apply multi-scale pooling and power transforms before classifying them with linear SVMs. By extracting rich representations from pixels and avoiding collapsing them prematurely, Sparse Code Gradients effectively learn how to measure local contrasts and find contours. We improve the F-measure metric on the BSDS500 benchmark to **0.74** (up from $0.71$ of gPb contours). Moreover, our learning approach can easily adapt to novel sensor data such as Kinect-style RGB-D cameras: Sparse Code Gradients on depth maps and surface normals lead to promising contour detection using depth and depth+color, as verified on the NYU Depth Dataset.

## 1 Introduction

Contour detection is a fundamental problem in vision. Accurately finding both object boundaries and interior contours has far reaching implications for many vision tasks including segmentation, recognition and scene understanding. High-quality image segmentation has increasingly been relying on contour analysis, such as in the widely used system of *Global Pb* [2]. Contours and segmentations have also seen extensive uses in shape matching and object recognition [8, 9].

Accurately finding contours in natural images is a challenging problem and has been extensively studied. With the availability of datasets with human-marked groundtruth contours, a variety of approaches have been proposed and evaluated (see a summary in [2]), such as learning to classify [17, 20, 16], contour grouping [23, 31, 12], multi-scale features [21, 2], and hierarchical region analysis [2]. Most of these approaches have one thing in common [17, 23, 31, 21, 12, 2]: they are built on top of a set of gradient features [17] measuring local contrast of oriented discs, using chi-square distances of histograms of color and textons. Despite various efforts to use generic image features [5] or learn them [16], these hand-designed gradients are still widely used after a decade and support top-ranking algorithms on the Berkeley benchmarks [2].

In this work, we demonstrate that contour detection can be vastly improved by replacing the hand-designed Pb gradients of [17] with rich representations that are automatically learned from data. We use sparse coding, in particularly Orthogonal Matching Pursuit [18] and K-SVD [1], to learn such representations on patches. Instead of a direct classification of patches [16], the sparse codes on the pixels are pooled over multi-scale half-discs for each orientation, in the spirit of the Pb

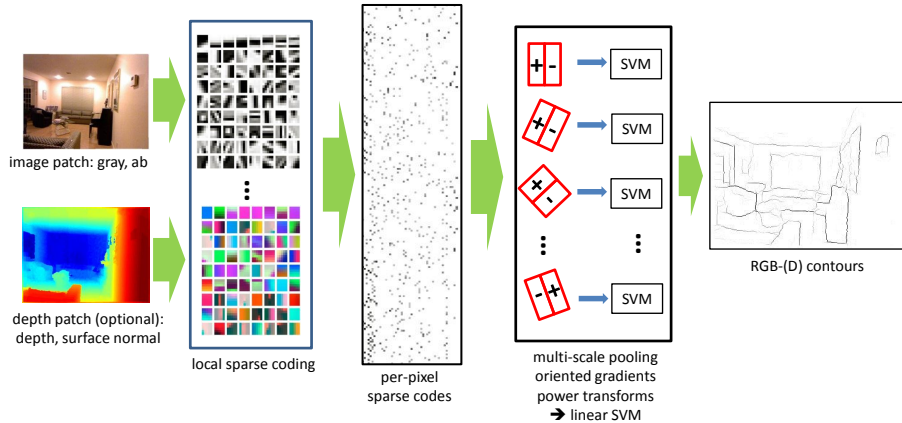

Figure 1: We combine sparse coding and oriented gradients for contour analysis on color as well as depth images. Sparse coding automatically learns a rich representation of patches from data. With multi-scale pooling, oriented gradients efficiently capture local contrast and lead to much more accurate contour detection than those using hand-designed features including *Global Pb* (gPb) [2].

gradients, before being classified with a linear SVM. The SVM outputs are then smoothed and non-max suppressed over orientations, as commonly done, to produce the final contours (see Fig. 1).

Our sparse code gradients (SCG) are much more effective in capturing local contour contrast than existing features. By only changing local features and keeping the smoothing and globalization parts fixed, we improve the F-measure on the BSDS500 benchmark to **0.74** (up from 0.71 of gPb), a substantial step toward human-level accuracy (see the precision-recall curves in Fig. 4). Large improvements in accuracy are also observed on other datasets including MSRC2 and PASCAL2008. Moreover, our approach is built on unsupervised feature learning and can directly apply to novel sensor data such as RGB-D images from Kinect-style depth cameras. Using the NYU Depth dataset [27], we verify that our SCG approach combines the strengths of color and depth contour detection and outperforms an adaptation of gPb to RGB-D by a large margin.

## 2 Related Work

Contour detection has a long history in computer vision as a fundamental building block. Modern approaches to contour detection are evaluated on datasets of natural images against human-marked groundtruth. The Pb work of Martin et. al. [17] combined a set of gradient features, using brightness, color and textons, to outperform the Canny edge detector on the Berkeley Benchmark (BSDS). Multi-scale versions of Pb were developed and found beneficial [21, 2]. Building on top of the Pb gradients, many approaches studied the globalization aspects, i.e. moving beyond local classification and enforcing consistency and continuity of contours. Ren et. al. developed CRF models on superpixels to learn junction types [23]. Zhu et. al. used circular embedding to enforce orderings of edgels [31]. The gPb work of Arbelaez et. al. computed gradients on eigenvectors of the affinity graph and combined them with local cues [2]. In addition to Pb gradients, Dollar et. al. [5] learned boosted trees on generic features such as gradients and Haar wavelets, Kokkinos used SIFT features on edgels [12], and Prasad et. al. [20] used raw pixels in class-specific settings. One closely related work was the discriminative sparse models of Mairal et al [16], which used K-SVD to represent multi-scale patches and had moderate success on the BSDS. A major difference of our work is the use of oriented gradients: comparing to directly classifying a patch, measuring contrast between oriented half-discs is a much easier problem and can be effectively learned.

Sparse coding represents a signal by reconstructing it using a small set of basis functions. It has seen wide uses in vision, for example for faces [28] and recognition [29]. Similar to deep network approaches [11, 14], recent works tried to avoid feature engineering and employed sparse coding of image patches to learn features from "scratch", for texture analysis [15] and object recognition [30, 3]. In particular, Orthogonal Matching Pursuit [18] is a greedy algorithm that incrementally finds sparse codes, and K-SVD is also efficient and popular for dictionary learning. Closely related to our work but on the different problem of recognition, Bo et. al. used matching pursuit and K-SVD to learn features in a coding hierarchy [3] and are extending their approach to RGB-D data [4].

Thanks to the mass production of Kinect, active RGB-D cameras became affordable and were quickly adopted in vision research and applications. The Kinect pose estimation of Shotton et. al. used random forests to learn from a huge amount of data [25]. Henry et. al. used RGB-D cameras to scan large environments into 3D models [10]. RGB-D data were also studied in the context of object recognition [13] and scene labeling [27, 22]. In-depth studies of contour and segmentation problems for depth data are much in need given the fast growing interests in RGB-D perception.

## 3 Contour Detection using Sparse Code Gradients

We start by examining the processing pipeline of *Global Pb* (gPb) [2], a highly influential and widely used system for contour detection. The gPb contour detection has two stages: local contrast estimation at multiple scales, and globalization of the local cues using spectral grouping. The core of the approach lies within its use of local cues in oriented gradients. Originally developed in [17], this set of features use relatively simple pixel representations (histograms of brightness, color and textons) and similarity functions (chi-square distance, manually chosen), comparing to recent advances in using rich representations for high-level recognition (e.g. [11, 29, 30, 3]).

We set out to show that both the pixel representation and the aggregation of pixel information in local neighborhoods can be much improved and, to a large extent, learned from and adapted to input data. For pixel representation, in Section 3.1 we show how to use *Orthogonal Matching Pursuit* [18] and *K-SVD* [1], efficient sparse coding and dictionary learning algorithms that readily apply to low-level vision, to extract sparse codes at every pixel. This sparse coding approach can be viewed similar in spirit to the use of filterbanks but avoids manual choices and thus directly applies to the RGB-D data from Kinect. We show learned dictionaries for a number of channels that exhibit different characteristics: grayscale/luminance, chromaticity (**ab**), depth, and surface normal.

In Section 3.2 we show how the pixel-level sparse codes can be integrated through multi-scale pooling into a rich representation of oriented local neighborhoods. By computing oriented gradients on this high dimensional representation and using a double power transform to code the features for linear classification, we show a linear SVM can be efficiently and effectively trained for each orientation to classify contour vs non-contour, yielding local contrast estimates that are much more accurate than the hand-designed features in gPb.

### 3.1 Local Sparse Representation of RGB-(D) Patches

**K-SVD and Orthogonal Matching Pursuit**. K-SVD [1] is a popular dictionary learning algorithm that generalizes K-Means and learns dictionaries of codewords from unsupervised data. Given a set of image patches $Y = [y_1, \cdots, y_n]$, K-SVD jointly finds a dictionary $D = [d_1, \cdots, d_m]$ and an associated sparse code matrix $X = [x_1, \cdots, x_n]$ by minimizing the reconstruction error

$$\min_{D,X} \|Y - DX\|_F^2 \quad s.t. \ \forall i, \ \|x_i\|_0 \leq K; \ \forall j, \ \|d_j\|_2 = 1 \quad (1)$$

where $\|\cdot\|_F$ denotes the Frobenius norm, $x_i$ are the columns of $X$, the zero-norm $\|\cdot\|_0$ counts the non-zero entries in the sparse code $x_i$, and $K$ is a predefined sparsity level (number of non-zero entries). This optimization can be solved in an alternating manner. Given the dictionary $D$, optimizing the sparse code matrix $X$ can be decoupled to sub-problems, each solved with Orthogonal Matching Pursuit (OMP) [18], a greedy algorithm for finding sparse codes. Given the codes $X$, the dictionary $D$ and its associated sparse coefficients are updated sequentially by singular value decomposition. For our purpose of representing local patches, the dictionary $D$ has a small size (we use 75 for 5x5 patches) and does not require a lot of sample patches, and it can be learned in a matter of minutes.

Once the dictionary $D$ is learned, we again use the Orthogonal Matching Pursuit (OMP) algorithm to compute sparse codes at every pixel. This can be efficiently done with convolution and a batch version of the OMP algorithm [24]. For a typical BSDS image of resolution 321x481, the sparse code extraction is efficient and takes 1∼2 seconds.

**Sparse Representation of RGB-D Data.** One advantage of unsupervised dictionary learning is that it readily applies to novel sensor data, such as the color and depth frames from a Kinect-style RGB-D camera. We learn K-SVD dictionaries up to four channels of color and depth: grayscale for luminance, chromaticity **ab** for color in the Lab space, depth (distance to camera) and surface normal (3-dim). The learned dictionaries are visualized in Fig. 2. These dictionaries are interesting

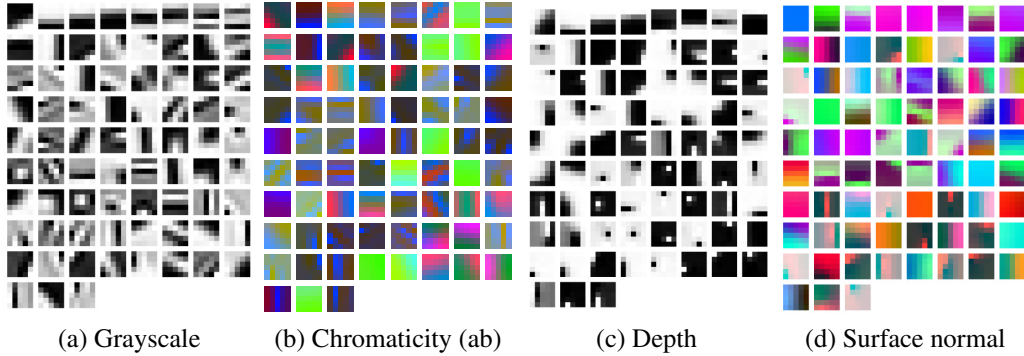

| (a) Grayscale | (b) Chromaticity (ab) | (c) Depth | (d) Surface normal |

Figure 2: K-SVD dictionaries learned for four different channels: grayscale and chromaticity (in **ab**) for an RGB image (a,b), and depth and surface normal for a depth image (c,d). We use a fixed dictionary size of 75 on 5x5 patches. The **ab** channel is visualized using a constant luminance of $50$. The 3-dimensional surface normal (xyz) is visualized in RGB (i.e. blue for frontal-parallel surfaces).

to look at and qualitatively distinctive: for example, the surface normal codewords tend to be more smooth due to flat surfaces, the depth codewords are also more smooth but with speckles, and the chromaticity codewords respect the opponent color pairs. The channels are coded separately.

### 3.2 Coding Multi-Scale Neighborhoods for Measuring Contrast

**Multi-Scale Pooling over Oriented Half-Discs.** Over decades of research on contour detection and related topics, a number of fundamental observations have been made, repeatedly: (1) contrast is the key to differentiate contour vs non-contour; (2) orientation is important for respecting contour continuity; and (3) multi-scale is useful. We do not wish to throw out these principles. Instead, we seek to adopt these principles for our case of high dimensional representations with sparse codes.

Each pixel is presented with sparse codes extracted from a small patch (5-by-5) around it. To aggregate pixel information, we use oriented half-discs as used in gPb (see an illustration in Fig. 1). Each orientation is processed separately. For each orientation, at each pixel $p$ and scale $s$, we define two half-discs (rectangles) $N^a$ and $N^b$ of size $s$-by-$(2s+1)$, on both sides of $p$, rotated to that orientation. For each half-disc $N$, we use average pooling on *non-zero* entries (i.e. a hybrid of average and max pooling) to generate its representation

$$F(N) = \left[ \sum_{i \in N} |x_{i1}| \middle/ \sum_{i \in N} I_{|x_{i1}|>0}, \cdots, \sum_{i \in N} |x_{im}| \middle/ \sum_{i \in N} I_{|x_{im}|>0} \right] \qquad (2)$$

where $x_{ij}$ is the $j$-th entry of the sparse code $x_i$, and $I$ is the indicator function whether $x_{ij}$ is non-zero. We rotate the image (after sparse coding) and use integral images for fast computations (on both $|x_{ij}|$ and $|x_{ij}| > 0$, whose costs are independent of the size of $N$.

For two oriented half-dics $N^a$ and $N^b$ at a scale $s$, we compute a difference (gradient) vector $D$

$$D(N_s^a, N_s^b) = \left| F(N_s^a) - F(N_s^b) \right| \qquad (3)$$

where $|\cdot|$ is an element-wise absolute value operation. We divide $D(N_s^a, N_s^b)$ by their norms $\|F(N_s^a)\| + \|F(N_s^b)\| + \epsilon$, where $\epsilon$ is a positive number. Since the magnitude of sparse codes varies over a wide range due to local variations in illumination as well as occlusion, this step makes the appearance features robust to such variations and increases their discriminative power, as commonly done in both contour detection and object recognition. This value is not hard to set, and we find a value of $\epsilon = 0.5$ is better than, for instance, $\epsilon = 0$.

At this stage, one could train a classifier on $D$ for each scale to convert it to a scalar value of contrast, which would resemble the chi-square distance function in gPb. Instead, we find that it is much better to avoid doing so separately at each scale, but combining multi-scale features in a joint representation, so as to allow interactions both between codewords and between scales. That is, our final representation of the contrast at a pixel $p$ is the concatenation of sparse codes pooled at all the

scales $s \in \{1, \cdots, S\}$ (we use $S = 4$):

$$D_p = \left[ D(N_1^a, N_1^b), \cdots, D(N_S^a, N_S^b); \ F(N_1^a \cup N_1^b), \cdots, F(N_S^a \cup N_S^b) \right] \qquad (4)$$

In addition to difference $D$, we also include a union term $F(N_s^a \cup N_s^b)$, which captures the appearance of the whole disc (union of the two half discs) and is normalized by $\|F(N_s^a)\| + \|F(N_s^b)\| + \epsilon$.

**Double Power Transform and Linear Classifiers.** The concatenated feature $D_p$ (non-negative) provides multi-scale contrast information for classifying whether $p$ is a contour location for a particular orientation. As $D_p$ is high dimensional (1200 and above in our experiments) and we need to do it at every pixel and every orientation, we prefer using linear SVMs for both efficient testing as well as training. Directly learning a linear function on $D_p$, however, does not work very well. Instead, we apply a double power transformation to make the features more suitable for linear SVMs

$$\overline{D}_p = \left[ D_p^{\alpha_1}, D_p^{\alpha_2} \right] \qquad (5)$$

where $0 < \alpha_1 < \alpha_2 < 1$. Empirically, we find that the double power transform works much better than either no transform or a single power transform $\alpha$, as sometimes done in other classification contexts. Perronnin et. al. [19] provided an intuition why a power transform helps classification, which "re-normalizes" the distribution of the features into a more Gaussian form. One plausible intuition for a double power transform is that the optimal exponent $\alpha$ may be different across feature dimensions. By putting two power transforms of $D_p$ together, we allow the classifier to pick its linear combination, different for each dimension, during the stage of supervised training.

**From Local Contrast to Global Contours.** We intentionally only change the local contrast estimation in gPb and keep the other steps fixed. These steps include: (1) the Savitzky-Goley filter to smooth responses and find peak locations; (2) non-max suppression over orientations; and (3) optionally, we apply the globalization step in gPb that computes a spectral gradient from the local gradients and then linearly combines the spectral gradient with the local ones. A sigmoid transform step is needed to convert the SVM outputs on $\overline{D}_p$ before computing spectral gradients.

## 4 Experiments

We use the evaluation framework of, and extensively compare to, the publicly available *Global Pb* (gPb) system [2], widely used as the state of the art for contour detection[1]. All the results reported on gPb are from running the gPb contour detection and evaluation codes (with default parameters), and accuracies are verified against the published results in [2]. The gPb evaluation includes a number of criteria, including precision-recall (P/R) curves from contour matching (Fig. 4), F-measures computed from P/R (Table 1,2,3) with a fixed contour threshold (ODS) or per-image thresholds (OIS), as well as average precisions (AP) from the P/R curves.

**Benchmark Datasets.** The main dataset we use is the BSDS500 benchmark [2], an extension of the original BSDS300 benchmark and commonly used for contour evaluation. It includes 500 natural images of roughly resolution 321x481, including 200 for training, 100 for validation, and 200 for testing. We conduct both color and grayscale experiments (where we convert the BSDS500 images to grayscale and retain the groundtruth). In addition, we also use the MSRC2 and PASCAL2008 segmentation datasets [26, 6], as done in the gPb work [2]. The MSRC2 dataset has 591 images of resolution 200x300; we randomly choose half for training and half for testing. The PASCAL2008 dataset includes 1023 images in its training and validation sets, roughly of resolution 350x500. We randomly choose half for training and half for testing.

For RGB-D contour detection, we use the NYU Depth dataset (v2) [27], which includes 1449 pairs of color and depth frames of resolution 480x640, with groundtruth semantic regions. We choose 60% images for training and 40% for testing, as in its scene labeling setup. The Kinect images are of lower quality than BSDS, and we resize the frames to 240x320 in our experiments.

**Training Sparse Code Gradients.** Given sparse codes from K-SVD and Orthogonal Matching Pursuit, we train the Sparse Code Gradients classifiers, one linear SVM per orientation, from sampled locations. For positive data, we sample groundtruth contour locations and estimate the orientations at these locations using groundtruth. For negative data, locations and orientations are random. We subtract the mean from the patches in each data channel. For BSDS500, we typically have 1.5 to 2

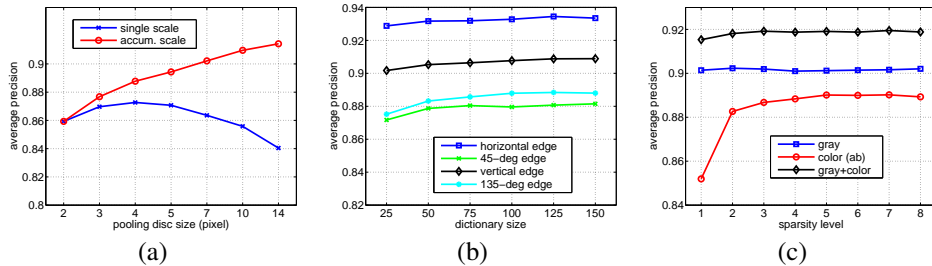

|     | (a)                         | (b)                         | (c)                         |

Figure 3: Analysis of our sparse code gradients, using average precision of classification on sampled boundaries. (a) The effect of single-scale vs multi-scale pooling (accumulated from the smallest). (b) Accuracy increasing with dictionary size, for four orientation channels. (c) The effect of the sparsity level K, which exhibits different behavior for grayscale and chromaticity.

|        |              | BSDS500 | | |
|--------|--------------|-----|-----|-----|
|        |              | ODS | OIS | AP  |
| local  | gPb (gray)   | .67 | .69 | .68 |
|        | SCG (gray)   | **.69** | **.71** | **.71** |
|        | gPb (color)  | .70 | .72 | .71 |
|        | SCG (color)  | **.72** | **.74** | **.75** |
| global | gPb (gray)   | .69 | .71 | .67 |
|        | SCG (gray)   | **.71** | **.73** | **.74** |
|        | gPb (color)  | .71 | .74 | .72 |
|        | SCG (color)  | **.74** | **.76** | **.77** |

Table 1: F-measure evaluation on the BSDS500 benchmark [2], comparing to gPb on grayscale and color images, both for local contour detection as well as for global detection (i.e. combined with the spectral gradient analysis in [2]).

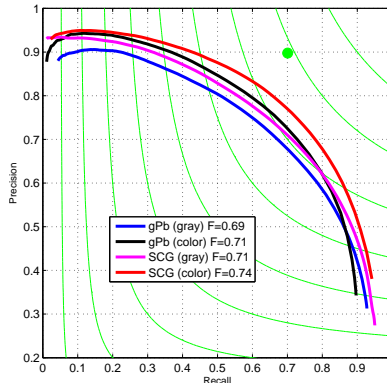

Figure 4: Precision-recall curves of SCG vs gPb on BSDS500, for grayscale and color images. We make a substantial step beyond the current state of the art toward reaching human-level accuracy (green dot).

million data points. We use 4 spatial scales, at half-disc sizes $2, 4, 7, 25$. For a dictionary size of 75 and 4 scales, the feature length for one data channel is 1200. For full RGB-D data, the dimension is 4800. For BSDS500, we train only using the 200 training images. We modify liblinear [7] to take dense matrices (features are dense after pooling) and single-precision floats.

**Looking under the Hood.** We empirically analyze a number of settings in our Sparse Code Gradients. In particular, we want to understand how the choices in the local sparse coding affect contour classification. Fig. 3 shows the effects of multi-scale pooling, dictionary size, and sparsity level (K). The numbers reported are intermediate results, namely the mean of average precision of four oriented gradient classifier $(0, 45, 90, 135$ degrees) on sampled locations (grayscale unless otherwise noted, on validation). As a reference, the average precision of gPb on this task is $0.878$.

For multi-scale pooling, the single best scale for the half-disc filter is about 4x8, consistent with the settings in gPb. For accumulated scales (using all the scales from the smallest up to the current level), the accuracy continues to increase and does not seem to be saturated, suggesting the use of larger scales. The dictionary size has a minor impact, and there is a small (yet observable) benefit to use dictionaries larger than 75, particularly for diagonal orientations (45- and 135-deg). The sparsity level $K$ is a more intriguing issue. In Fig. 3(c), we see that for grayscale only, $K = 1$ (normalized nearest neighbor) does quite well; on the other hand, color needs a larger $K$, possibly because **ab** is a nonlinear space. When combining grayscale and color, it seems that we want $K$ to be at least 3. It also varies with orientation: horizontal and vertical edges require a smaller $K$ than diagonal edges. (If using $K = 1$, our final F-measure on BSDS500 is $0.730$.)

We also empirically evaluate the double power transform vs single power transform vs no transform. With no transform, the average precision is $0.865$. With a single power transform, the best choice of the exponent is around $0.4$, with average precision $0.884$. A double power transform (with exponents

| | MSRC2 | | |
|---|---|---|---|
| | ODS | OIS | AP |
| gPb | .37 | .39 | .22 |
| SCG | **.43** | **.43** | **.33** |
| | PASCAL2008 | | |
| | ODS | OIS | AP |
| gPb | .34 | .38 | .20 |
| SCG | **.37** | **.41** | **.27** |

| | RGB-D (NYU v2) | | |
|---|---|---|---|
| | ODS | OIS | AP |
| gPb (color) | .51 | .52 | .37 |
| SCG (color) | **.55** | **.57** | **.46** |
| gPb (depth) | .44 | .46 | .28 |
| SCG (depth) | **.53** | **.54** | **.45** |
| gPb (RGB-D) | .53 | .54 | .40 |
| SCG (RGB-D) | **.62** | **.63** | **.54** |

Table 2: F-measure evaluation comparing our SCG approach to gPb on two additional image datasets with contour groundtruth: MSRC2 [26] and PASCAL2008 [6].

Table 3: F-measure evaluation on RGB-D contour detection using the NYU dataset (v2) [27]. We compare to gPb on using color image only, depth only, as well as color+depth.

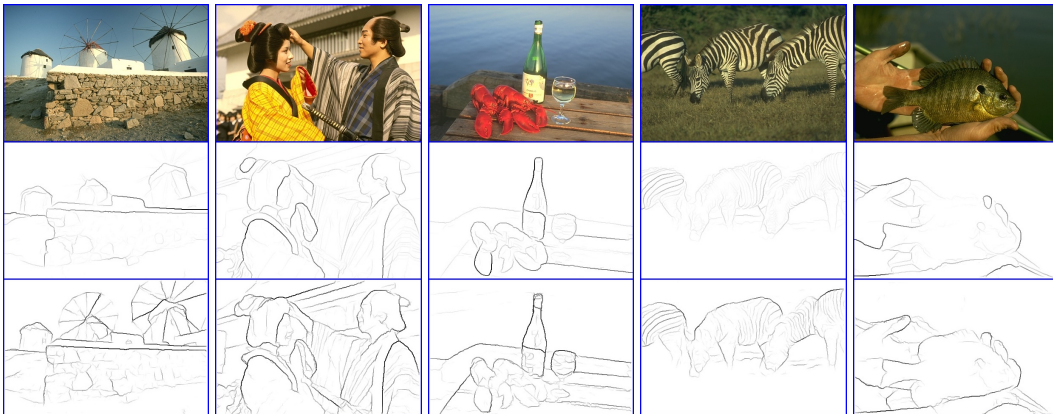

Figure 5: Examples from the BSDS500 dataset [2]. (Top) Image; (Middle) gPb output; (Bottom) SCG output (this work). Our SCG operator learns to preserve fine details (e.g. windmills, faces, fish fins) while at the same time achieving higher precision on large-scale contours (e.g. back of zebras). (Contours are shown in double width for the sake of visualization.)

0.25 and 0.75, which can be computed through sqrt) improves the average precision to 0.900, which translates to a large improvement in contour detection accuracy.

**Image Benchmarking Results.** In Table 1 and Fig. 4 we show the precision-recall of our Sparse Code Gradients vs gPb on the BSDS500 benchmark. We conduct four sets of experiments, using color or grayscale images, with or without the globalization component (for which we use exactly the same setup as in gPb). Using Sparse Code Gradients leads to a significant improvement in accuracy in all four cases. The local version of our SCG operator, i.e. only using local contrast, is already better ($F = 0.72$) than gPb with globalization ($F = 0.71$). The full version, local SCG plus spectral gradient (computed from local SCG), reaches an F-measure of **0.739**, a large step forward from gPb, as seen in the precision-recall curves in Fig. 4. On BSDS300, our F-measure is $0.715$.

We observe that SCG seems to pick up fine-scale details much better than gPb, hence the much higher recall rate, while maintaining higher precision over the entire range. This can be seen in the examples shown in Fig. 5. While our scale range is similar to that of gPb, the multi-scale pooling scheme allows the flexibility of learning the balance of scales separately for each code word, which may help detecting the details. The supplemental material contains more comparison examples.

In Table 2 we show the benchmarking results for two additional datasets, MSRC2 and PAS-CAL2008. Again we observe large improvements in accuracy, in spite of the somewhat different natures of the scenes in these datasets. The improvement on MSRC2 is much larger, partly because the images are smaller, hence the contours are smaller in scale and may be over-smoothed in gPb.

As for computational cost, using integral images, local SCG takes ~100 seconds to compute on a single-thread Intel Core i5-2500 CPU on a BSDS image. It is slower than but comparable to the highly optimized multi-thread C++ implementation of gPb (~60 seconds).

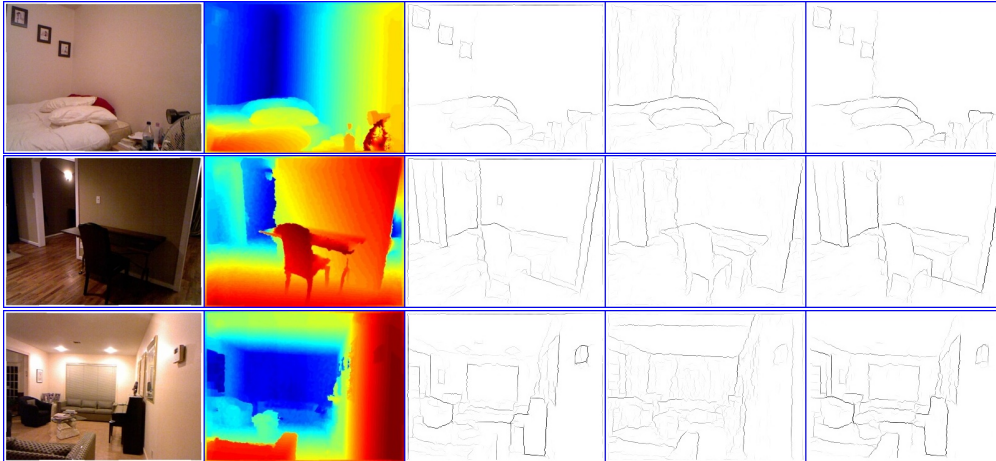

Figure 6: Examples of RGB-D contour detection on the NYU dataset (v2) [27]. The five panels are: input image, input depth, image-only contours, depth-only contours, and color+depth contours. Color is good picking up details such as photos on the wall, and depth is useful where color is uniform (e.g. corner of a room, row 1) or illumination is poor (e.g. chair, row 2).

**RGB-D Contour Detection.** We use the second version of the NYU Depth Dataset [27], which has higher quality groundtruth than the first version. A median filtering is applied to remove double contours (boundaries from two adjacent regions) within 3 pixels. For RGB-D baseline, we use a simple adaptation of gPb: the depth values are in meters and used directly as a grayscale image in gPb gradient computation. We use a linear combination to put (soft) color and depth gradients together in gPb before non-max suppression, with the weight set from validation.

Table 3 lists the precision-recall evaluations of SCG vs gPb for RGB-D contour detection. All the SCG settings (such as scales and dictionary sizes) are kept the same as for BSDS. SCG again outperforms gPb in all the cases. In particular, we are much better for depth-only contours, for which gPb is not designed. Our approach learns the low-level representations of depth data fully automatically and does not require any manual tweaking. We also achieve a much larger boost by combining color and depth, demonstrating that color and depth channels contain complementary information and are both critical for RGB-D contour detection. Qualitatively, it is easy to see that RGB-D combines the strengths of color and depth and is a promising direction for contour and segmentation tasks and indoor scene analysis in general [22]. Fig. 6 shows a few examples of RGB-D contours from our SCG operator. There are plenty of such cases where color alone or depth alone would fail to extract contours for meaningful parts of the scenes, and color+depth would succeed.

## 5 Discussions

In this work we successfully showed how to learn and code local representations to extract contours in natural images. Our approach combined the proven concept of oriented gradients with powerful representations that are automatically learned through sparse coding. Sparse Code Gradients (SCG) performed significantly better than hand-designed features that were in use for a decade, and pushed contour detection much closer to human-level accuracy as illustrated on the BSDS500 benchmark.

Comparing to hand-designed features (e.g. Global Pb [2]), we maintain the high dimensional representation from pooling oriented neighborhoods and do not collapse them prematurely (such as computing chi-square distance at each scale). This passes a richer set of information into learning contour classification, where a double power transform effectively codes the features for linear SVMs. Comparing to previous learning approaches (e.g. discriminative dictionaries in [16]), our uses of multi-scale pooling and oriented gradients lead to much higher classification accuracies.

Our work opens up future possibilities for learning contour detection and segmentation. As we illustrated, there is a lot of information locally that is waiting to be extracted, and a learning approach such as sparse coding provides a principled way to do so, where rich representations can be automatically constructed and adapted. This is particularly important for novel sensor data such as RGB-D, for which we have less understanding but increasingly more need.

## Footnotes

[1]In this work we focus on contour detection and do not address how to derive segmentations from contours.

# References

[1] M. Aharon, M. Elad, and A. Bruckstein. K-SVD: An algorithm for designing overcomplete dictionaries for sparse representation. *IEEE Transactions on Signal Processing*, 54(11):4311–4322, 2006.

[2] P. Arbelaez, M. Maire, C. Fowlkes, and J. Malik. Contour detection and hierarchical image segmentation. *IEEE Trans. PAMI*, 33(5):898–916, 2011.

[3] L. Bo, X. Ren, and D. Fox. Hierarchical Matching Pursuit for Image Classification: Architecture and Fast Algorithms. In *Advances in Neural Information Processing Systems 24*, 2011.

[4] L. Bo, X. Ren, and D. Fox. Unsupervised Feature Learning for RGB-D Based Object Recognition. In *International Symposium on Experimental Robotics (ISER)*, 2012.

[5] P. Dollar, Z. Tu, and S. Belongie. Supervised learning of edges and object boundaries. In *CVPR*, volume 2, pages 1964–71, 2006.

[6] M. Everingham, L. Van Gool, C. K. I. Williams, J. Winn, and A. Zisserman. The PASCAL Visual Object Classes Challenge 2008 (VOC2008). http://www.pascal-network.org/challenges/VOC/voc2008/.

[7] R. Fan, K. Chang, C. Hsieh, X. Wang, and C. Lin. Liblinear: A library for large linear classification. *The Journal of Machine Learning Research*, 9:1871–1874, 2008.

[8] V. Ferrari, T. Tuytelaars, and L. V. Gool. Object detection by contour segment networks. In *ECCV*, pages 14–28, 2006.

[9] C. Gu, J. Lim, P. Arbeláez, and J. Malik. Recognition using regions. In *CVPR*, pages 1030–1037, 2009.

[10] P. Henry, M. Krainin, E. Herbst, X. Ren, and D. Fox. Rgb-d mapping: Using depth cameras for dense 3d modeling of indoor environments. In *International Symposium on Experimental Robotics (ISER)*, 2010.

[11] G. Hinton, S. Osindero, and Y. Teh. A fast learning algorithm for deep belief nets. *Neural computation*, 18(7):1527–1554, 2006.

[12] I. Kokkinos. Highly accurate boundary detection and grouping. In *CVPR*, pages 2520–2527, 2010.

[13] K. Lai, L. Bo, X. Ren, and D. Fox. A large-scale hierarchical multi-view RGB-D object dataset. In *ICRA*, pages 1817–1824, 2011.

[14] H. Lee, R. Grosse, R. Ranganath, and A. Ng. Convolutional deep belief networks for scalable unsupervised learning of hierarchical representations. In *ICML*, pages 609–616, 2009.

[15] J. Mairal, F. Bach, J. Ponce, G. Sapiro, and A. Zisserman. Discriminative learned dictionaries for local image analysis. In *CVPR*, pages 1–8, 2008.

[16] J. Mairal, M. Leordeanu, F. Bach, M. Hebert, and J. Ponce. Discriminative sparse image models for class-specific edge detection and image interpretation. *ECCV*, pages 43–56, 2008.

[17] D. Martin, C. Fowlkes, and J. Malik. Learning to detect natural image boundaries using brightness and texture. In *Advances in Neural Information Processing Systems 15*, 2002.

[18] Y. Pati, R. Rezaiifar, and P. Krishnaprasad. Orthogonal Matching Pursuit: Recursive Function Approximation with Applications to Wavelet Decomposition. In *The Twenty-Seventh Asilomar Conference on Signals, Systems and Computers*, pages 40–44, 1993.

[19] F. Perronnin, J. Sánchez, and T. Mensink. Improving the fisher kernel for large-scale image classification. In *ECCV*, pages 143–156, 2010.

[20] M. Prasad, A. Zisserman, A. Fitzgibbon, M. Kumar, and P. Torr. Learning class-specific edges for object detection and segmentation. *Computer Vision, Graphics and Image Processing*, pages 94–105, 2006.

[21] X. Ren. Multi-scale improves boundary detection in natural images. In *ECCV*, pages 533–545, 2008.

[22] X. Ren, L. Bo, and D. Fox. RGB-(D) scene labeling: features and algorithms. In *Computer Vision and Pattern Recognition (CVPR), 2012 IEEE Conference on*, pages 2759–2766. IEEE, 2012.

[23] X. Ren, C. Fowlkes, and J. Malik. Cue integration in figure/ground labeling. In *Advances in Neural Information Processing Systems 18*, 2005.

[24] R. Rubinstein, M. Zibulevsky, and M. Elad. Efficient Implementation of the K-SVD Algorithm using Batch Orthogonal Matching Pursuit. Technical report, CS Technion, 2008.

[25] J. Shotton, A. Fitzgibbon, M. Cook, T. Sharp, M. Finocchio, R. Moore, A. Kipman, and A. Blake. Real-time human pose recognition in parts from single depth images. In *CVPR*, volume 2, page 3, 2011.

[26] J. Shotton, J. Winn, C. Rother, and A. Criminisi. Textonboost: Joint appearance, shape and context modeling for multi-class object recognition and segmentation. In *ECCV*, 2006.

[27] N. Silberman and R. Fergus. Indoor scene segmentation using a structured light sensor. In *IEEE Workshop on 3D Representation and Recognition (3dRR)*, 2011.

[28] J. Wright, A. Yang, A. Ganesh, S. Sastry, and Y. Ma. Robust face recognition via sparse representation. *IEEE Trans. PAMI*, 31(2):210–227, 2009.

[29] J. Yang, K. Yu, Y. Gong, and T. Huang. Linear spatial pyramid matching using sparse coding for image classification. In *CVPR*, pages 1794–1801, 2009.

[30] K. Yu, Y. Lin, and J. Lafferty. Learning image representations from the pixel level via hierarchical sparse coding. In *CVPR*, pages 1713–1720, 2011.

[31] Q. Zhu, G. Song, and J. Shi. Untangling cycles for contour grouping. In *ICCV*, 2007.

